# The 'Moving Targets' Training Algorithm

**Richard Rohwer**
Centre for Speech Technology Research
Edinburgh University
80, South Bridge
Edinburgh EH1 1HN SCOTLAND

## ABSTRACT

A simple method for training the dynamical behavior of a neural network is derived. It is applicable to any training problem in discrete-time networks with arbitrary feedback. The algorithm resembles back-propagation in that an error function is minimized using a gradient-based method, but the optimization is carried out in the hidden part of state space either instead of, or in addition to weight space. Computational results are presented for some simple dynamical training problems, one of which requires response to a signal 100 time steps in the past.

## 1   INTRODUCTION

This paper presents a minimization-based algorithm for training the dynamical behavior of a discrete-time neural network model. The central idea is to treat hidden nodes as target nodes with variable training data. These "moving targets" are varied during the minimization process. Werbos (Werbos, 1983) used the term "moving targets" to describe the qualitative idea that a network should set itself intermediate objectives, and vary these objectives as information is accumulated on their attainability and their usefulness for achieving overall objectives. The (coincidentally) like-named algorithm presented here can be regarded as a quantitative realization of this qualitative idea.

The literature contains several temporal training algorithms based on minimization of an error measure with respect to the weights. This type of method includes the straightforward extension of the back-propagation method to back-propagation

through time (Rumelhart, 1986), the methods of Rohwer and Forrest (Rohwer, 1987), Pearlmutter (Pearlmutter, 1989), and the forward propagation of derivatives (Robinson, 1988, Williams 1989a, Williams 1989b, Kuhn, 1990). A careful comparison of moving targets with back-propagation in time and teacher forcing appears in (Rohwer, 1989b). Although applicable only to fixed-point training, the algorithms of Almeida (Almeida, 1989) and Pineda (Pineda, 1988) have much in common with these dynamical training algorithms. The formal relationship between these and the method of Rohwer and Forrest is spelled out in (Rohwer 1989a).

## 2   NOTATION AND STATEMENT OF THE TRAINING PROBLEM

Consider a neural network model with arbitrary feedback as a dynamical system in which the dynamical variables $x_{it}$ change with time according to a dynamical law given by the mapping

$$\left. \begin{array}{rcl} x_{it} & = & \sum_j w_{ij} f(x_{j,t-1}) \quad i > 0 \\ x_{0t} & = & \text{bias constant} \end{array} \right\} \tag{1}$$

unless specified otherwise. The *weights* $w_{ij}$ are arbitrary parameters representing the connection strength from node $j$ to node $i$. $f$ is an arbitrary differentiable function. Let us call any given variable $x_{it}$ the *"activation"* on node $i$ at time $t$. It represents the total input into node $i$ at time $t$. Let the *"output"* of each node be denoted by $y_{it} = f(x_{it})$. Let node 0 be a *"bias node"*, assigned a positive constant activation so that the weights $w_{i0}$ can be interpreted as activation thresholds.

In normal back-propagation, a network architecture is defined which divides the network into input, hidden, and target nodes. The moving targets algorithm makes itself applicable to arbitrary training problems by defining analogous concepts in a manner dependent upon the training data, but independent of the network architecture. Let us call a node-time pair an *"event"*. To define a training problem, the set of all events must be divided into three disjoint sets, the *input* events $I$, *target* events $T$, and *hidden* events $H$. A node may participate in different types of event at different times. For every input event $(it) \in I$, we require *training data* $X_{it}$ with which to overrule the dynamical law (1) using

$$x_{it} = X_{it} \quad (it) \in I. \tag{2}$$

(The bias events $(0t)$ can be regarded as a special case of input events.) For each target event $(it) \in T$, we require training data $X_{it}$ to specify a desired activation value for event $(0t)$. No notational ambiguity arises from referring to input and target data with the same symbol $X$ because $I$ and $T$ are required to be disjoint sets. The training data says nothing about the hidden events in $H$. There is no restriction on how the initial events $(i0)$ are classified.

## 3   THE "MOVING TARGETS" METHOD

Like back-propagation, the moving targets training method uses (arbitrary) gradient-based minimization techniques to minimize an *"error"* function such as the *"output deficit"*

$$E_{\text{od}} = \tfrac{1}{2} \sum_{(it)\in T} \{y_{it} - Y_{it}\}^2,  \tag{3}$$

where $y_{it} = f(x_{it})$ and $Y_{it} = f(X_{it})$. A modification of the output deficit error gave the best results in numerical experiments. However, the most elegant formalism follows from an *"activation deficit"* error function:

$$E_{\text{ad}} = \tfrac{1}{2} \sum_{(it)\in T} \{x_{it} - X_{it}\}^2,  \tag{4}$$

so this is what we shall use to present the formalism.

The basic idea is to treat the hidden node activations as variable target activations. Therefore let us denote these variables as $X_{it}$, just as the (fixed) targets and inputs are denoted. Let us write the computed activation values $x_{it}$ of the hidden and target events in terms of the inputs and (fixed and moving) targets of the previous time step. Then let us extend the sum in (4) to include the hidden events, so the error becomes

$$E = \tfrac{1}{2} \sum_{(it)\in T\cup H} \left\{ \sum_j w_{ij} f(X_{j,t-1}) - X_{it} \right\}^2.  \tag{5}$$

This is a function of the weights $w_{ij}$, and because there are no $x$'s present, the full dependence on $w_{ij}$ is explicitly displayed. We do not actually have desired values for the $X_{it}$ with $(it) \in H$. But any values for which weights can be found which make (5) vanish would be suitable, because this would imply not only that the desired targets are attained, but also that the dynamical law is followed on both the hidden and target nodes. Therefore let us regard E as a function of both the weights and the "moving targets" $X_{it}, (it) \in H$. This is the essence of the method. The derivatives with respect to all of the independent variables can be computed and plugged into a standard minimization algorithm.

The reason for preferring the activation deficit form of the error (4) to the output deficit form (3) is that the activation deficit form makes (5) purely quadratic in the weights. Therefore the equations for the minimum,

$$dE/dw_{ij} = \partial E/\partial w_{ij} = 0,  \tag{6}$$

form a linear system, the solution of which provides the optimal weights for any given set of moving targets. Therefore these equations might as well be used to define the weights as functions of the moving targets, thereby making the error (5) a *function of the moving targets alone.*

The derivation of the derivatives with respect to the moving targets is spelled out in (Rohwer, 1989b). The result is:

$$\frac{dE}{dX_{as}} = \sum_i \chi_{i,s+1} e_{i,s+1} w_{ia} f'_{as} - \chi_{as} e_{as}, \tag{7}$$

where

$$\chi_{it} = \begin{cases} 1 & (it) \in T \cup H \\ 0 & (it) \notin T \cup H \end{cases} \tag{8}$$

$$e_{it} = \sum_j w_{ij} f(X_{j,t-1}) - X_{it}, \tag{9}$$

$$f'_{it} = \frac{df(x)}{dx}\Bigg|_{x=X_{it}}, \tag{10}$$

and

$$w_{ij} = \sum_k \left( \sum_t \chi_{it} X_{it} Y_{k,t-1} \right) M_{kj}^{(i)-1}, \tag{11}$$

where $M^{(a)-1}$ is the inverse of $M^{(a)}$, the correlation matrix of the node outputs defined by

$$M_{ij}^{(a)} = \sum_t \chi_{at} Y_{i,t-1} Y_{j,t-1}. \tag{12}$$

In the event that any of the matrices $M$ are singular, a pseudo-inversion method such as singular value decomposition (Press, 1988) can be used to define a unique solution among the infinite number available.

Note also that (11) calls for a separate matrix inversion for each node. However if the set of input nodes remains fixed for all time, then all these matrices are equal.

## 3.1  FEEDFORWARD VERSION

The basic ideas used in the moving targets algorithm can be applied to feedforward networks to provide an alternative method to back-propagation. The hidden node activations for each training example become the moving target variables. Further details appear in (Rohwer, 1989b). The moving targets method for feedforward nets is analogous to the method of Grossman, Meir, and Domany (Grossman, 1990a, 1990b) for networks with discrete node values. Birmiwal, Sarwal, and Sinha (Birmiwal, 1989) have developed an algorithm for feedforward networks which incorporates the use of hidden node values as fundamental variables and a linear

system of equations for obtaining the weight matrix. Their algorithm differs from
the feedforward version of moving targets mainly in the (inessential) use of a specific
minimization algorithm which discards most of the gradient information except for
the signs of the various derivatives. Heileman, Georgiopoulos, and Brown (Heile-
man, 1989) also have an algorithm which bears some resemblance to the feedforward
version of moving targets. Another similar algorithm has been developed by Krogh,
Hertz, and Thorbergasson (Krogh, 1989, 1990).

## 4    COMPUTATIONAL RESULTS

A set of numerical experiments performed with the activation deficit form of the
algorithm (4) is reported in (Rohwer, 1989b). Some success was attained, but
greater progress was made after changing to a quartic output deficit error function
with temporal weighting of errors:

$$E_{\text{quartic}} = \tfrac{1}{4} \sum_{(it)\in T} (1.0 + at)\{y_{it} - Y_{it}\}^4. \tag{13}$$

Here $a$ is a small positive constant. The quartic function is dominated by the terms
with the greatest error. This combats a tendency to fail on a few infrequently seen
state transitions in order to gain unneeded accuracy on a large number of similar,
low-error state transitions. The temporal weighting encourages the algorithm to
focus first on late-time errors, and then work back in time. In some cases this
helped with local minimum difficulties. A difficulty with convergence to chaotic
attractors reported in (Rohwer, 1989b) appears to have mysteriously disappeared
with the adoption of this error measure.

### 4.1    MINIMIZATION ALGORITHM

Further progress was made by altering the minimization algorithm. Originally the
conjugate gradient algorithm (Press, 1988) was used, with a linesearch algorithm
from Fletcher (Fletcher, 1980). The new algorithm might be called "curvature
avoidance". The change in the gradient with each linesearch is used to update
a moving average estimate of the absolute value of the diagonal components of
the Hessian. The linesearch direction is taken to be the component-by-component
quotient of the gradient with these curvature averages. Were it not for the absolute
values, this would be an unusual way of estimating the conjugate gradient. The
absolute values are used to discourage exploration of directions which show any
hint of being highly curved. The philosophy is that by exploring low-curvature
directions first, narrow canyons are entered only when necessary.

### 4.2    SIMULATIONS

Several simulations have been done using fully connected networks. Figure 1 plots
the node outputs of a network trained to switch between different limit cycles under
input control. There are two input nodes, one target node, and 2 hidden nodes,
as indicated in the left margin. Time proceeds from left to right. The oscillation

period of the target node increases with the binary number represented by the two input nodes. The network was trained on one period of each of the four frequencies.

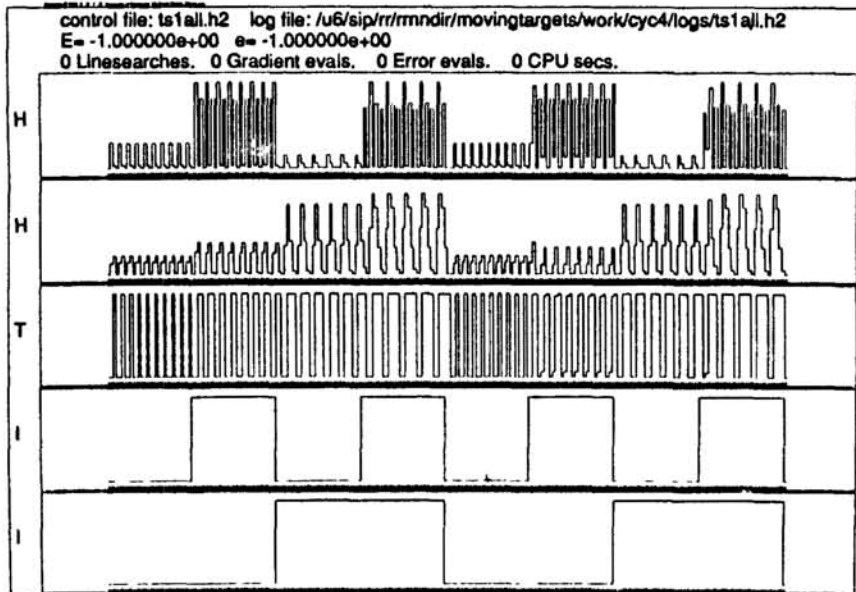

**Figure 1:** Controlled switching between limit cycles

Figure 2 shows the operation of a network trained to detect whether an even or odd number of pulses have been presented to the input; a temporal version of parity detection. The network was trained on the data preceding the third input pulse.

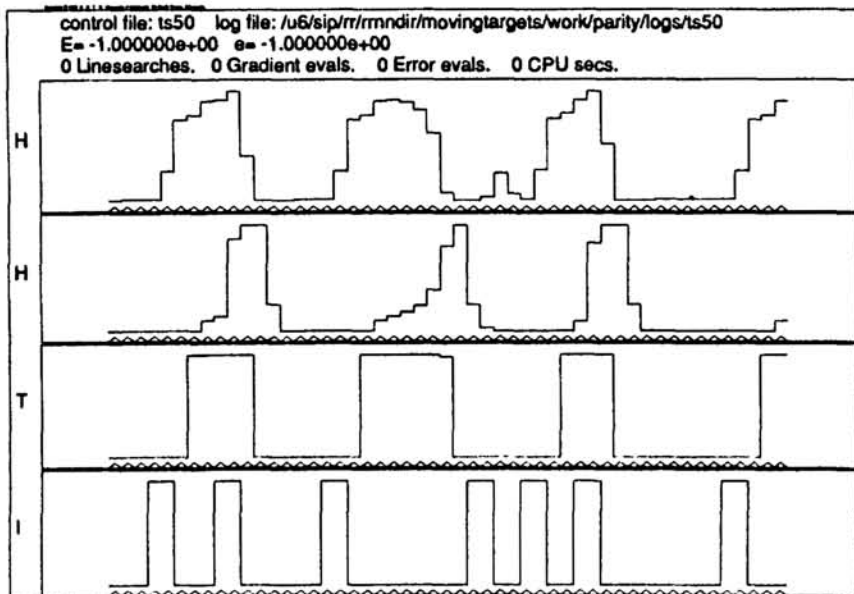

**Figure 2:** Parity detection

Figure 3 shows the behavior of a network trained to respond to the second of two input pulses separated by 100 time steps. This demonstrates a unique (in the author's knowledge) capability of this method, an ability to utilize very distant

temporal correlations when there is no other way to solve the problem. This network was trained and tested on the same data, the point being merely to show that training is possible in this type of problem. More complex problems of this type frequently get stuck in local minima.

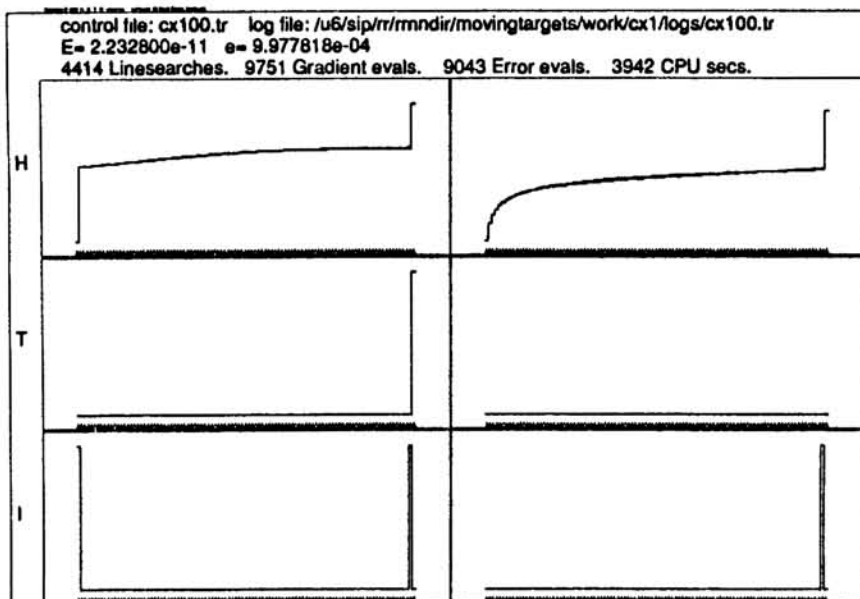

**Figure 3:** Responding to temporally distant input

# 5    CONCLUDING REMARKS

The simulations show that this method works, and show in particular that distant temporal correlations can be discovered. Some practical difficulties have emerged, however, which are currently limiting the application of this technique to 'toy' problems. The most serious are local minima and long training times. Problems involving large amounts of training data may present the minimization problem with an impractically large number of variables. Variations of the algorithm are being studied in hopes of overcomming these difficulties.

**Acknowledgements**

This work was supported by ESPRIT Basic Research Action 3207 ACTS.

**References**

L. Almeida, (1989), "Backpropagation in Non-Feedforward Networks", in *Neural Computing Architectures*, I. Aleksander, ed., North Oxford Academic.

K. Birmiwal, P. Sarwal, and S. Sinha, (1989), "A new Gradient-Free Learning Algorithm", Tech. report, Dept. of EE, Southern Illinois U., Carbondale.

R. Fletcher, (1980), *Practical Methods of Optimization*, v1, Wiley.

T. Grossman, (1990a), "The CHIR Algorithm: A Generalization for Multiple Output and Multilayered Networks", to appear in *Complex Systems*.

T. Grossman, (1990b), this volume.

G. L. Heileman, M. Georgiopoulos, and A. K. Brown, (1989), "The Minimal Disturbance Back Propagation Algorithm", Tech. report, Dept. of EE, U. of Central Florida, Orlando.

A. Krogh, J. A. Hertz, and G. I. Thorbergsson, (1989), "A Cost Function for Internal Representations", NORDITA preprint 89/37 S.

A. Krogh, J. A. Hertz, and G. I. Thorbergsson, (1990), this volume.

G. Kuhn, (1990) "Connected Recognition with a Recurrent Network", to appear in Proc. NEUROSPEECH, 18 May 1989, as special issue of *Speech Communication*, **9**, no. 2.

B. Pearlmutter, (1989), "Learning State Space Trajectories in Recurrent Neural Networks", *Proc. IEEE IJCNN 89*, Washington D. C., II-365.

F. Pineda, (1988), "Dynamics and Architecture for Neural Computation", *J. Complexity* **4**, 216.

W. H. Press, B. P. Flannery, S. A. Teukolsky, and W. T. Vetterling, (1988), *Numerical Recipes in C, The Art of Scientific Computing*, Cambridge.

A. J. Robinson and F. Fallside, (1988), "Static and Dynamic Error Propagation Networks with Applications to Speech Coding", *Neural Information Processing Systems*, D. Z. Anderson, Ed., AIP, New York.

R. Rohwer and B. Forrest, (1987), "Training Time Dependence in Neural Networks" *Proc. IEEE ICNN*, San Diego, II-701.

R. Rohwer and S. Renals, (1989a), "Training Recurrent Networks", in *Neural Networks from Models to Applications*, L. Personnaz and G. Dreyfus, eds., I.D.S.E.T., Paris, 207.

R. Rohwer, (1989b), "The 'Moving Targets' Training Algorithm", to appear in *Proc. DANIP*, GMD Bonn, J. Kinderman and A. Linden, Eds.

D. Rumelhart, G. Hinton and R. Williams, (1986), "Learning Internal Representations by Error Propagation" in *Parallel Distributed Processing*, v. 1, MIT.

P. Werbos, (1983) *Energy Models and Studies*, B. Lev, Ed., North Holland.

R. Williams and D. Zipser, (1989a), "A Learning Algorithm for Continually Running Fully Recurrent Neural Networks", *Neural Computation* **1**, 270.

R. Williams and D. Zipser, (1989b), "Experimental Analysis of the Real-time Recurrent Learning Algorithm", *Connection Science* **1**, 87.